# Margin-Based Algorithms
# for Information Filtering*

**Nicolò Cesa-Bianchi**
DTI, University of Milan
via Bramante 65
26013 Crema, Italy
*cesa-bianchi@dti.unimi.it*

**Alex Conconi**
DTI, University of Milan
via Bramante 65
26013 Crema, Italy
*conconi@dti.unimi.it*

**Claudio Gentile**
CRII, Università dell'Insubria
Via Ravasi, 2
21100 Varese, Italy
*gentile@dsi.unimi.it*

## Abstract

In this work, we study an information filtering model where the relevance labels associated to a sequence of feature vectors are realizations of an unknown probabilistic linear function. Building on the analysis of a restricted version of our model, we derive a general filtering rule based on the margin of a ridge regression estimator. While our rule may observe the label of a vector only by classfying the vector as relevant, experiments on a real-world document filtering problem show that the performance of our rule is close to that of the on-line classifier which is allowed to observe all labels. These empirical results are complemented by a theoretical analysis where we consider a randomized variant of our rule and prove that its expected number of mistakes is never much larger than that of the optimal filtering rule which knows the hidden linear model.

## 1  Introduction

Systems able to filter out unwanted pieces of information are of crucial importance for several applications. Consider a stream of discrete data that are individually labelled as "relevant" or "nonrelevant" according to some fixed relevance criterion; for instance, news about a certain topic, emails that are not spam, or fraud cases from logged data of user behavior. In all of these cases, a filter can be used to drop uninteresting parts of the stream, forwarding to the user only those data which are likely to fulfil the relevance criterion. From this point of view, the filter may be viewed as a simple on-line binary classifier. However, unlike standard on-line pattern classification tasks, where the classifier observes the correct label after each prediction, here the relevance of a data element is known only if the filter decides to forward that data element to the user. This learning protocol with partial feedback is known as *adaptive filtering* in the Information Retrieval community (see, e.g., [14]). We formalize the filtering problem as follows. Each element of an arbitrary data sequence is characterized by a feature vector $x \in \mathbb{R}^N$ and an associated relevance label $y$ (say, $y = +1$ for relevant and $y = -1$ for nonrelevant). At each time $t = 1, 2, \ldots$, the filtering system observes the $t$-th feature vector $x_t$ and must decide whether or not to forward it. If the data is forwarded, then its relevance label $y_t$ is revealed to the system,

which may use this information to adapt the filtering criterion. If the data is not forwarded, its relevance label remains hidden. We call $\boldsymbol{x}_t$ the $t$-th *instance* of the data sequence and the pair $(\boldsymbol{x}_t, y_t)$ the $t$-th example. For simplicity, we assume $\|\boldsymbol{x}_t\| = 1$ for all $t \geq 1$. There are two kinds of errors the filtering system can make in judging the relevance of a feature vector $\boldsymbol{x}_t$. We say that an example $(\boldsymbol{x}_t, y_t)$ is a *false positive* if $y_t = -1$ and $\boldsymbol{x}_t$ is classified as relevant by the system; similarly, we say that $(\boldsymbol{x}_t, y_t)$ is a *false negative* if $y_t = +1$ and $\boldsymbol{x}_t$ is classified as nonrelevant by the system. Although false negatives remain unknown, the filtering system is scored according to the overall number of wrong relevance judgements it makes. That is, *both* false positives and false negatives are counted as mistakes. In this paper, we study the filtering problem under the assumption that relevance judgements are generated using an unknown probabilistic linear function. We design filtering rules that maintain a linear hypothesis and use the margin information to decide whether to forward the next instance. Our performance measure is the regret; i.e., the number of wrong judgements made by a filtering rule over and above those made by the rule knowing the probabilistic function used to generate judgements. We show finite-time (nonasymptotical) bounds on the regret that hold for arbitrary sequences of instances. The only other results of this kind we are aware of are those proven in [9] for the *apple tasting* model. Since in the apple tasting model relevance judgements are chosen adversarially rather than probabilistically, we cannot compare their bounds with ours. We report some preliminary experimental results which might suggest the superiority of our methods as opposed to the general transformations developed within the apple tasting framework. As a matter of fact, these general transformations do not take margin information into account.

In Section 2, we introduce our probabilistic relevance model and make some preliminary observations. In Section 3, we consider a restricted version of the model within which we prove a regret bound for a simple filtering rule called SIMPLE-FIL. In Section 4, we generalize this filtering rule and show its good performance on the Reuters Corpus Volume 1. The algorithm employed, which we call RIDGE-FIL, is a linear least squares algorithm inspired by [2]. In that section we also prove, within the unrestricted probabilistic model, a regret bound for the randomized variant P-RIDGE-FIL of the general filtering rule. Both RIDGE-FIL and its randomized variant can be run with kernels [13] and adapted to the case when the unknown linear function drifts with time.

## 2 Learning model, notational conventions and preliminaries

The relevance of $\boldsymbol{x}_t$ is given by a $\{-1, 1\}$-valued random variable $Y_t$ (where $Y_t = 1$ means "relevant") such that there exists a fixed and unknown vector $\boldsymbol{u} \in \mathbb{R}^N$, $\|\boldsymbol{u}\| = 1$, for which $\mathbb{E}[Y_t] = \boldsymbol{u}^\top \boldsymbol{x}_t$ for all $t = 1, 2, \dots, n$. Hence $\boldsymbol{x}_t$ is relevant with probability $(1 + \boldsymbol{u}^\top \boldsymbol{x}_t)/2 \in [0, 1]$. The random variables $Y_1, \dots, Y_n$ are assumed to be independent, whereas we do not make any assumption on the way the sequence $\boldsymbol{x}_1, \boldsymbol{x}_2, \dots, \boldsymbol{x}_n$ is generated. In this model, we want to perform almost as well as the algorithm that knows $\boldsymbol{u}$ and forwards $\boldsymbol{x}_t$ if and only if $\boldsymbol{u}^\top \boldsymbol{x}_t \geq 0$. We consider linear-threshold filtering algorithms that predict the value of $Y_t$ through $\mathrm{SGN}\left(\boldsymbol{w}_t^\top \boldsymbol{x}_t - \tau_t\right)$, where $\boldsymbol{w}_t \in \mathbb{R}^N$ is a dynamically updated weight vector which might be intended as the current approximation to $\boldsymbol{u}$, and $\tau_t$ is a suitable time-changing "confidence" threshold. For any fixed sequence $\boldsymbol{x}_1, \boldsymbol{x}_2, \dots, \boldsymbol{x}_n \in \mathbb{R}^N$ of instances, we use $\Delta_t$ to denote the margin $\boldsymbol{u}^\top \boldsymbol{x}_t$ and $\widehat{\Delta}_t$ to denote the margin $\boldsymbol{w}_t^\top \boldsymbol{x}_t$. We define the *expected regret* of the linear-threshold filtering algorithm at time $t$ as $\mathbb{P}(Y_t(\widehat{\Delta}_t - \tau_t) < 0) - \mathbb{P}(Y_t \Delta_t < 0)$. We observe that in the conditional $\mathbb{P}'$-probability space where $\widehat{\Delta}_t - \tau_t$ is given we have

$$
\begin{aligned}
&\mathbb{P}'(Y_t(\widehat{\Delta}_t - \tau_t) < 0) - \mathbb{P}'(Y_t \Delta_t < 0) \\
={}& (\mathbb{P}'(Y_t(\widehat{\Delta}_t - \tau_t) < 0) - \mathbb{P}'(Y_t \Delta_t < 0))\{(\widehat{\Delta}_t - \tau_t)\Delta_t \leq 0\} \\
={}& (\mathbb{P}'(Y_t \Delta_t \geq 0) - \mathbb{P}'(Y_t \Delta_t < 0))\left\{(\widehat{\Delta}_t - \tau_t)\Delta_t \leq 0\right\} = |\Delta_t|\left\{(\widehat{\Delta}_t - \tau_t)\Delta_t \leq 0\right\},
\end{aligned}
$$

where we use $\{\phi\}$ to denote the Bernoulli random variable which is 1 if and only if predicate $\phi$ is true. Integrating over all possible values of $\widehat{\Delta}_t - \tau_t$ we obtain

$$
\begin{aligned}
\mathbb{P}(Y_t(\widehat{\Delta}_t - \tau_t) < 0) - \mathbb{P}(Y_t \Delta_t < 0) &= |\Delta_t| \, \mathbb{P}((\widehat{\Delta}_t - \tau_t)\Delta_t \leq 0) & (1) \\
&\leq |\Delta_t| \, \mathbb{P}(|\widehat{\Delta}_t - \tau_t - \Delta_t| \geq |\Delta_t|) & \\
&\leq \tfrac{1}{|\Delta_t|} E[(\widehat{\Delta}_t - \tau_t - \Delta_t)^2], & (2)
\end{aligned}
$$

where the last inequality is Markov's. These (in)equalities will be used in Sections 3 and 4.2 for the analysis of SIMPLE-FIL and P-RIDGE-FIL algorithms.

## 3  A simplified model

We start by analyzing a restricted model where each data element has the same unknown probability $p$ of being relevant and we want to perform almost as well as the filtering rule that consistently does the optimal action (i.e., always forwards if $p \geq 1/2$ and never forwards otherwise). The analysis of this model is used in Section 4 to guide the design of good filtering rules for the unrestricted model.

Let $\delta = p - 1/2$ and let $\widehat{\delta}_{N_t} = \widehat{p}_{N_t} - 1/2$ be the sample average of $\delta$, where $N_t$ is the number of forwarded data elements in the first $t$ time steps and $\widehat{p}_{N_t}$ is the fraction of true positives among the $N_t$ elements that have been forwarded. Obviously, the optimal rule forwards if and only if $\delta \geq 0$. Consider instead the empirical rule that forwards if and only if $\widehat{\delta}_{N_{t-1}} \geq 0$. This rule makes a mistake only when $\delta \widehat{\delta}_{N_{t-1}} \leq 0$. To make the probability of this event go to zero with $t$, it suffices that $\mathbb{P}(|\widehat{\delta}_{N_t} - \delta| \leq |\delta|) \to 1$ as $t \to \infty$, which can only happen if $N_t$ increases quickly enough with $t$. Hence, data should be forwarded (irrespective to the sign of the estimate $\widehat{\delta}_{N_t}$) also when the confidence level for $\widehat{\delta}_{N_t}$ gets too small with respect to $t$. A problem in this argument is that large deviation bounds require $N_t = \Omega(1/\delta^2)$ for making $\mathbb{P}(|\widehat{\delta}_{N_t} - \delta| > |\delta|)$ small. But in our case $\delta^2$ is unknown. To fix this, we use the condition $N_t = \Omega(1/\widehat{\delta}_{N_t}^2)$. This looks dangerous, as we use the empirical value of $\widehat{\delta}_{N_t}$ to control the large deviations of $\widehat{\delta}_{N_t}$ itself; however, we will show that this approach indeed works. An algorithm, which we call SIMPLE-FIL, implementing the above line of reasoning takes the form of the following simple rule: forward if and only if $\widehat{\delta}_{N_{t-1}} - \tau_t \geq 0$, where $\tau_t = -\sqrt{4(\ln t)/N_{t-1}}$. The expected regret at time $t$ of SIMPLE-FIL is defined as the probability that SIMPLE-FIL makes a mistake at time $t$ minus the probability that the optimal filtering rule makes a mistake, that is $\mathbb{P}(Y_t(\widehat{\delta}_{N_{t-1}} - \tau_t) < 0) - \mathbb{P}(Y_t \delta < 0)$. The next result shows a logarithmic bound on this regret.

**Theorem 1** *The expected cumulative regret of* SIMPLE-FIL *after any number $n \geq 1$ of time steps is at most* $(32/|\delta| + 2) \ln n + 4$.

*Proof sketch.* We can bound the actual regret after $n$ time steps as follows. From (1) and the definition of the filtering rule we have

$$
\begin{aligned}
\sum_{t=1}^n \mathbb{P}(Y_t(\widehat{\delta}_{N_{t-1}} - \tau_t) < 0) - \sum_{t=1}^n \mathbb{P}(Y_t \delta < 0) &= 2|\delta| \sum_{t=1}^n \mathbb{P}\left\{(\widehat{\delta}_{N_{t-1}} - \tau_t)\delta \leq 0\right\} \\
&\leq 2|\delta| \left(\sum_{t=1}^n \mathbb{P}\left\{N_{t-1} \leq \frac{4\ln t}{\widehat{\delta}_{N_{t-1}}^2}\right\} + \sum_{t=1}^n \mathbb{P}\left\{\widehat{\delta}_{N_{t-1}}\delta \leq 0, \, N_{t-1} > \frac{4\ln t}{\widehat{\delta}_{N_{t-1}}^2}\right\}\right) \\
&= 2|\delta| \left(\mathbb{E}[Z_A] + \mathbb{E}[Z_B]\right).
\end{aligned}
$$

Without loss of generality, assume $n > 1$. We now bound $\mathbb{E}[Z_A]$ and $\mathbb{E}[Z_B]$ separately. Since $N_{t-1} \leq 4(\ln t)/\widehat{\delta}^2_{N_{t-1}}$ implies that $N_t = N_{t-1} + 1$, we have that $Z_A \geq \ell$ implies $N_s \geq \ell$ for some $s \geq \ell$. Hence we can write

$$
\begin{aligned}
Z_A &\leq \ell + \sum_{t=\ell+1}^{n} \left\{ N_{t-1} \leq \frac{4\ln t}{\widehat{\delta}^2_{N_{t-1}}}, N_{t-1} \geq \ell \right\} \leq \ell + \sum_{t=\ell+1}^{n} \sum_{s=\ell}^{t-1} \left\{ s \leq \frac{4\ln t}{\widehat{\delta}^2_s} \right\} \\
&\leq \ell + \sum_{t=\ell+1}^{n} \sum_{s=\ell}^{t-1} \left( \left\{ s \leq \frac{16\ln t}{\delta^2} \right\} + \left\{ |\widehat{\delta}_s| \leq |\delta|/2 \right\} \right) \\
&\leq \ell + \sum_{t=\ell+1}^{n} \sum_{s=\ell}^{t-1} \left\{ |\widehat{\delta}_s| \leq |\delta|/2 \right\} \quad \text{for } \ell \geq (16\ln n)/\delta^2 + 1 \\
&\leq \ell + \sum_{t=\ell+1}^{n} \sum_{s=\ell}^{t-1} \left\{ |\widehat{\delta}_s - \delta| \geq |\delta|/2 \right\} .
\end{aligned}
$$

Applying Chernoff-Hoeffding [11] bounds to $\widehat{\delta}_s$, which is a sum of $s$ $\{-1/2, 1/2\}$-valued independent random variables, we obtain

$$
\mathbb{E}[Z_A] \leq \ell + \sum_{t=\ell+1}^{n} \sum_{s=\ell}^{t-1} \mathbb{P}\left( |\widehat{\delta}_s - \delta| \geq |\delta|/2 \right) \leq \frac{16\ln n}{\delta^2} + 2 .
$$

We now bound $\mathbb{E}[Z_B]$ by adapting a technique from [8]. Let $a_{s,t} = \sqrt{\frac{\ln t}{\delta^2 s}}$, $\widehat{a}_{s,t} = \sqrt{\frac{\ln t}{\widehat{\delta}^2_s s}}$, $\widehat{\varepsilon}_{s,t} = \frac{\widehat{a}_{s,t}}{1 - \widehat{a}_{s,t}}$. We have

$$
\begin{aligned}
Z_B &\leq \sum_{t=1}^{n} \left\{ |\widehat{\delta}_{N_{t-1}} - \delta| \geq a_{N_{t-1},t}|\delta| \right\} + \sum_{t=1}^{n} \left\{ a_{N_{t-1},t} \geq \widehat{\varepsilon}_{N_{t-1},t} \right\} \\
&\quad + \sum_{t=1}^{n} \left\{ \widehat{\varepsilon}_{N_{t-1},t} \geq 1, N_{t-1} > \frac{4\ln t}{\widehat{\delta}^2_{N_{t-1}}} \right\} \\
&\leq 2\sum_{t=1}^{n} \sum_{s=0}^{t-1} \left\{ |\widehat{\delta}_s - \delta| \geq a_{s,t}|\delta| \right\} + \sum_{t=1}^{n} \left\{ \widehat{\varepsilon}_{N_{t-1},t} \geq 1, N_{t-1} > \frac{4\ln t}{\widehat{\delta}^2_{N_{t-1}}} \right\} \\
&= Z_C + Z_D .
\end{aligned}
$$

Applying Chernoff-Hoeffding bounds again, we get $\mathbb{E}[Z_C] \leq 2\sum_{t=1}^{n} \frac{1}{t} \leq 2(1 + \ln n)$. Finally, one can easily verify that $Z_D = 0$. Piecing everything together we get the desired result. $\qquad\square$

## 4 Linear least squares filtering

In order to generalize SIMPLE-FIL to the original (unrestricted) learning model described in Section 2, we need a low-variance estimate of the target vector $\boldsymbol{u}$. Let $S_t$ be the matrix whose columns are the forwarded feature vectors after the first $t$ time steps and let $\boldsymbol{Y}_t$ be the vector of corresponding observed relevance labels (the index $t$ will be momentarily dropped). Note that $\mathbb{E}\boldsymbol{Y} = S^\top \boldsymbol{u}$ holds. Consider the least squares estimator $(S S^\top)^\dagger S \boldsymbol{Y}$ of $\boldsymbol{u}$, where $(S S^\top)^\dagger$ is the pseudo-inverse of $S S^\top$. For all $\boldsymbol{u}$ belonging to the column space of $S$, this is an unbiased estimator of $\boldsymbol{u}$, that is $\mathbb{E}\left[ (S S^\top)^\dagger S \boldsymbol{Y} \right] = (S S^\top)^\dagger S \mathbb{E}\boldsymbol{Y} = (S S^\top)^\dagger S S^\top \boldsymbol{u} = \boldsymbol{u}$. To remove the assumption on $\boldsymbol{u}$, we make $S S^\top$ full rank by adding the identity $I$. This also allows us to replace the pseudo-inverse with the standard inverse,

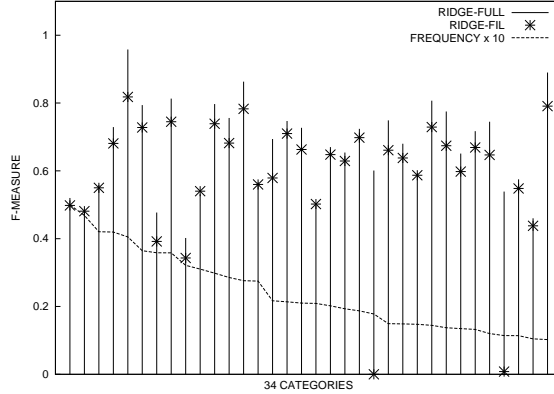

Figure 1: $F$-measure for each one of the 34 filtering tasks. The $F$-measure is defined by $2RP/(R+P)$, where $P$ is precision (fraction of relevant documents among the forwarded ones) and $R$ is recall (fraction of forwarded documents among the relevant ones). In the plot, the filtering rule RIDGE-FIL is compared with RIDGE-FULL which sees the correct label after each classification. While precision and recall of RIDGE-FULL are balanced, RIDGE-FIL's recall is higher than precision due to the need of forwarding more documents than believed relevant. This in order to make the confidence of the estimator converge to 1 fast enough. Note that, in some cases, this imbalance causes RIDGE-FIL to achieve a slightly better $F$-measure than RIDGE-FULL.

obtaining $(I + S\,S^\top)^{-1}S\,\boldsymbol{Y}$, a "sparse" version of the ridge regression estimator [12] (the sparsity is due to the fact that we only store in $S$ the forwarded instances, i.e., those for which we have a relevance labels). To estimate directly the margin $\boldsymbol{u}^\top\boldsymbol{x}$, rather than $\boldsymbol{u}$, we further modify, along the lines of the techniques analyzed in [3, 6, 15], the sparse ridge regression estimator. More precisely, we estimate $\boldsymbol{u}^\top\boldsymbol{x}_t$ with the quantity $\boldsymbol{w}_t^\top\boldsymbol{x}_t$, where the $\boldsymbol{w}_t$ is defined by

$$\boldsymbol{w}_t = (I + S_{t-1}\,S_{t-1}^\top + \boldsymbol{x}_t\,\boldsymbol{x}_t^\top)^{-1}S_{t-1}\,\boldsymbol{Y}_{t-1}. \tag{3}$$

Using the Sherman-Morrison formula, we can then write out the expectation of $\boldsymbol{w}_t^\top\boldsymbol{x}_t$ as
$\mathbb{E}\left[\boldsymbol{w}_t^\top\boldsymbol{x}_t\right] = \frac{\boldsymbol{u}^\top\boldsymbol{x}_t - \boldsymbol{u}^\top(I+S_{t-1}\,S_{t-1}^\top)^{-1}\boldsymbol{x}_t}{1+\boldsymbol{x}_t^\top(I+S_{t-1}\,S_{t-1}^\top)^{-1}\boldsymbol{x}_t}$, which holds for all $\boldsymbol{u}$, $\boldsymbol{x}_t$, and all matrices $S_{t-1}$. Let $N_{t-1}$ be the number of forwarded instances among $\boldsymbol{x}_1, \dots, \boldsymbol{x}_{t-1}$. In order to generalize to the estimator (3) the analysis of SIMPLE-FIL, we need to find a large deviation bound of the form $\mathbb{P}\left(\left|\boldsymbol{w}_t^\top\boldsymbol{x}_t - \Delta_t\right| > \varepsilon,\ N_{t-1} = s\right) = o(1)$, for all $\varepsilon > 0$, where $o(1)$ goes to zero "sufficiently fast" as $s \to \infty$. Though we have not been able to find such bounds, we report some experimental results showing that algorithms based on (3) and inspired by the analysis of SIMPLE-FIL do exhibit a good empirical behavior on real-world data. Moreover, in Section 4.2 we prove a bound (not based on the analysis of SIMPLE-FIL) on the expected regret of a randomized variant of the algorithm used in the experiments. For this variant we are able to prove a regret bound that scales essentially with the square root of $n$ (to be contrasted with the logarithmic regret of SIMPLE-FIL).

## 4.1 Experimental results

We ran our experiments using the filtering rule that forwards $\boldsymbol{x}_t$ if $\text{SGN}(\boldsymbol{w}_t^\top\boldsymbol{x}_t - \tau_t) = 1$, where $\boldsymbol{w}_t$ is the estimator (3) and $\tau_t = -\sqrt{(5\ln t)/N_{t-1}}$. Note that this rule, which we call RIDGE-FIL, is a natural generalization of SIMPLE-FIL to the unrestricted learning model; in particular, SIMPLE-FIL uses a relevance threshold $\tau_t$ of the very same form as RIDGE-FIL, although SIMPLE-FIL's "margin" $\widehat{\delta}$ is defined differently. We tested our algorithm on a

**Algorithm:** P-RIDGE-FIL.
**Parameters:** Real $a > 0$; $p \in [0, 1]$.
**Initialization:** $\boldsymbol{w}_1 = (0, \dots, 0)$, $A_1 = aI$ $S_1 = \emptyset$, $k = 1$.
Loop for $t = 1, 2, \dots$

    1. Get $\boldsymbol{x}_t \in \mathbb{R}^N$ and let $\widehat{\Delta}_t = \boldsymbol{w}_k^\top \boldsymbol{x}_t$.

    2. **If** $\widehat{\Delta}_t \geq 0$ **then** forward $\boldsymbol{x}_t$, get label $y_t$ and update as follows:
        $\boldsymbol{w}_k^m = \boldsymbol{w}_k + (y_t - \widehat{\Delta}_t) A_k^{-1} \boldsymbol{x}_t$;
        $S_{k+1} = [S_k \ \boldsymbol{x}_t]$; $A_{k+1} = S_{k+1} S_{k+1}^\top$;
        $\boldsymbol{w}_{k+1} = (A_{k+1} + \lambda I)^{-1} A_{k+1} \boldsymbol{w}_k^m$, where $\lambda = 0$ if $\|\boldsymbol{w}_k^m\| \leq 1$ and $\lambda > 0$ is such that $\|\boldsymbol{w}_{k+1}\| = 1$, otherwise;
        $k \leftarrow k + 1$.

    3. **Else** forward $\boldsymbol{x}_t$ with probability $p$. If $\boldsymbol{x}_t$ was forwarded then get label $y_t$ and do the same updates as in 2; otherwise, do not make any update.

Figure 2: Pseudo-code for the filtering algorithm P-RIDGE-FIL. The performance of this algorithm is analyzed in Theorem 3.

document filtering problem based on the first 70000 newswire stories from the Reuters Corpus Volume 1. We selected the 34 Reuters topics whose frequency in the set of 70000 documents was between 1% and 5% (a plausible range for filtering applications). For each topic, we defined a filtering task whose relevance judgements were assigned based on whether the document was labelled with that topic or not. Documents were mapped to real vectors using the bag-of-words representation. In particular, after tokenization we lemmatized the tokens using a general-purpose finite-state morphological English analyzer and then removed stopwords (we also replaced all digits with a single special character). Document vectors were built removing all words which did not occur at least three times in the corpus and using the TF-IDF encoding in the form $(1 + \ln \text{TF}) \ln(N/\text{DF})$, where TF is the word frequency in the document, DF is the number of documents containing the word, and $N$ is the total number of documents (if TF$= 0$ the TF-IDF coefficient was also set to 0). Finally, all document vectors were normalized to length 1. To measure how the choice of the threshold $\tau_t$ affects the filtering performance, we ran RIDGE-FIL with $\tau_t$ set to zero on the same dataset as a standard on-line binary classifier (i.e., receiving the correct label after every classification). We call this algorithm RIDGE-FULL. Figure 1 illustrates the experimental results. The average $F$-measure of RIDGE-FULL and RIDGE-FIL are, respectively, $0.68$ and $0.59$; hence the threshold compensates pretty well the partial feedback in the filtering setup. On the other hand, the standard Perceptron achieves here a $F$-measure of $0.61$ in the classification task, hence inferior to that of RIDGE-FULL. Finally, we also tested the apple-tasting filtering rule (see [9, STAP transformation]) based on the binary classifier RIDGE-FULL. This transformation, which does not take into consideration the margin, exhibited a poor performance and we did not include it in the plot.

## 4.2 Probabilistic ridge filtering

In this section we introduce a probabilistic filtering algorithm, derived from the (on-line) ridge regression algorithm, for the class of linear probabilistic relevance functions. The algorithm, called P-RIDGE-FIL, is sketched in Figure 2. The algorithm takes $a > 0$ and a probability value $p$ as input parameters and maintains a linear hypothesis $\boldsymbol{w}_k$. If $\boldsymbol{w}_k^\top \boldsymbol{x}_t \geq 0$, then $\boldsymbol{x}_t$ is forwarded and $\boldsymbol{w}_k$ gets updated according to the following two-steps ridge regression-like rule. First, the intermediate vector $\boldsymbol{w}_k^m$ is computed via the standard on-line ridge regression algorithm using the inverse of matrix $A_k$. Then, the new vector $\boldsymbol{w}_{k+1}$ is obtained by *projecting* $\boldsymbol{w}_k^m$ onto the unit ball, where the projection is taken w.r.t. the

distance function $d_{k+1}(\boldsymbol{u}, \boldsymbol{w}) = \frac{1}{2}(\boldsymbol{u} - \boldsymbol{w})^\top A_{k+1}(\boldsymbol{u} - \boldsymbol{w})$. Note that $||\boldsymbol{w}_k^m|| \leq 1$ implies $\boldsymbol{w}_{k+1} = \boldsymbol{w}_k^m$. On the other hand, if $\boldsymbol{w}_k^\top \boldsymbol{x}_t < 0$ then $\boldsymbol{x}_t$ is forwarded (and consequently $\boldsymbol{w}_k$ is updated) with some probability $p$. The analysis of P-RIDGE-FIL is inspired by the analysis in [1] for a related but different problem, and is based on relating the expected regret in a given trial $t$ to a measure of the progress of $\boldsymbol{w}_t$ towards $\boldsymbol{u}$. The following lemma will be useful.

**Lemma 2** *Using the notation of Figure 2, let $t$ be the trial when the $k$-th update occurs. Then the following inequality holds:* $\frac{1}{2}(\widehat{\Delta}_t - Y_t)^2 - \frac{1}{2}(\Delta_t - Y_t)^2 \leq 2\ln\frac{|A_{k+1}|}{|A_k|} + d_k(\boldsymbol{u}, \boldsymbol{w}_k) - d_{k+1}(\boldsymbol{u}, \boldsymbol{w}_{k+1})$, *where $|A|$ denotes the determinant of matrix $A$ and $d_k(\boldsymbol{u}, \boldsymbol{w}) = \frac{1}{2}(\boldsymbol{u} - \boldsymbol{w})^\top A_k(\boldsymbol{u} - \boldsymbol{w})$.*

*Proof sketch.* From Lemma 4.2 and Theorem 4.6 in [3] and the fact that $|\widehat{\Delta}_k| \leq ||\boldsymbol{w}_k|| \leq 1$ $\forall k$ it follows that $\frac{1}{2}(\widehat{\Delta}_t - Y_t)^2 - \frac{1}{2}(\Delta_t - Y_t)^2 \leq 2\ln\frac{|A_{k+1}|}{|A_k|} + d_k(\boldsymbol{u}, \boldsymbol{w}_k) - d_{k+1}(\boldsymbol{u}, \boldsymbol{w}_k^m)$. Now, the function $d_{k+1}(\boldsymbol{u}, \boldsymbol{w}) = \frac{1}{2}(\boldsymbol{u} - \boldsymbol{w})^\top A_{k+1}(\boldsymbol{u} - \boldsymbol{w})$ is a Bregman divergence (e.g., [4, 10]), and it can be easily shown that $\boldsymbol{w}_{k+1}$ in Figure 2 is the projection of $\boldsymbol{w}_k^m$ onto the convex set $\{\boldsymbol{v} \in \mathbb{R}^N : ||\boldsymbol{v}|| \leq 1\}$ w.r.t. $d_{k+1}$; i.e., $\boldsymbol{w}_{k+1} = \operatorname{argmin}_{\boldsymbol{v} \in \mathbb{R}^N : ||\boldsymbol{v}|| \leq 1} d_{k+1}(\boldsymbol{v}, \boldsymbol{w}_k^m)$. By a projection property of Bregman divergences (see, e.g., the appendix in [10]) it follows that $d_{k+1}(\boldsymbol{u}, \boldsymbol{w}_k^m) \geq d_{k+1}(\boldsymbol{u}, \boldsymbol{w}_{k+1})$ for all $\boldsymbol{u}$ such that $||\boldsymbol{u}|| \leq 1$. Putting together gives the desired inequality. $\square$

**Theorem 3** *Let $\gamma = \min_t |\Delta_t| = \min_t |\boldsymbol{u}^\top \boldsymbol{x}_t|$. For all $n > 1$, if algorithm P-RIDGE-FIL of Figure 2 is run with[1] $p = \frac{1}{2}\frac{1}{\sqrt{n}}(1 + \frac{1}{\sqrt{n}})$, then its expected cumulative regret $\sum_{t=1}^n \mathbb{P}(Y_t\widehat{\Delta}_t < 0) - \sum_{t=1}^n \mathbb{P}(Y_t\Delta_t < 0)$ is at most*

$$2a\frac{\sqrt{n}}{\gamma} + 8\frac{\sqrt{n}}{\gamma}N\ln\left(1 + \frac{n}{aN}\right) + \frac{1}{2}\sqrt{n} + \frac{1}{2}.$$

*Proof sketch.* If $t$ is the trial when the $k$-th forward takes place, we define the random variables $D_t = d_k(\boldsymbol{u}, \boldsymbol{w}_k) - d_{k+1}(\boldsymbol{u}, \boldsymbol{w}_{k+1})$ and $\beta_t = 2\ln\frac{|A_{k+1}|}{|A_k|}$. If no update occurs in trial $t$ we set $D_t = \beta_t = 0$. Let $R_t$ be the regret of P-RIDGE-FIL in trial $t$ and $R_t'$ be the regret of the update rule $\boldsymbol{w}_k \to \boldsymbol{w}_{k+1}$ in trial $t$. If $\widehat{\Delta}_t \geq 0$, then $\mathbb{E}[R_t] = \mathbb{E}[R_t']$ and $\mathbb{E}[D_t]$ can be lower bounded via Lemma 2. If $\widehat{\Delta}_t < 0$, then $\mathbb{E}[D_t]$ gets lower bounded via Lemma 2 only with probability $p$, while for the regret we can only use the trivial bound $\mathbb{E}[R_t] \leq 1$. With probability $1 - p$, instead, $\mathbb{E}[R_t] = \mathbb{E}[R_t']$ and $\mathbb{E}[D_t] = 0$. Let $\kappa$ be a constant to be specified. We can write

$$\begin{aligned}
\gamma\mathbb{E}[R_t] - \kappa\,\mathbb{E}[D_t] = {}&\gamma\mathbb{E}[R_t\{\widehat{\Delta}_t \geq 0\}] - \kappa\,\mathbb{E}[D_t\{\widehat{\Delta}_t \geq 0\}] \\
&+ \gamma\mathbb{E}[R_t\{\widehat{\Delta}_t < 0\}] - \kappa\,\mathbb{E}[D_t\{\widehat{\Delta}_t < 0\}].
\end{aligned} \tag{4}$$

Now, it is easy to verify that in the conditional space where $\widehat{\Delta}_t$ is given we have $\mathbb{E}[(\Delta_t - Y_t)^2 \mid \widehat{\Delta}_t] = 1 - \Delta_t^2$ and $\mathbb{E}[(\widehat{\Delta}_t - Y_t)^2 \mid \widehat{\Delta}_t] = (\widehat{\Delta}_t - \Delta_t)^2 + 1 - \Delta_t^2$. Thus, using Lemma 2 and Eq. (4) we can write

$$\begin{aligned}
\gamma\mathbb{E}[R_t] - \kappa\,\mathbb{E}[D_t] = {}&\gamma\,\mathbb{E}[R_t'\{\widehat{\Delta}_t \geq 0\}] - \kappa\,\mathbb{E}[(\tfrac{1}{2}(\widehat{\Delta}_t - \Delta_t)^2 - \mathbb{E}[\beta_t|\widehat{\Delta}_t])\{\widehat{\Delta}_t \geq 0\}] \\
&+ \gamma\left((1-p)\mathbb{E}[R_t'\{\widehat{\Delta}_t < 0\}] + p\mathbb{E}[1\{\widehat{\Delta}_t < 0\}]\right) \\
&- \kappa\,p\,\mathbb{E}[(\tfrac{1}{2}(\widehat{\Delta}_t - \Delta_t)^2 - \mathbb{E}[\beta_t|\widehat{\Delta}_t])\{\widehat{\Delta}_t < 0\}]
\end{aligned}$$

This can be further upper bounded by

$$\mathbb{E}[(\widehat{\Delta}_t - \Delta_t)^2] - \kappa\,\mathbb{E}[\tfrac{1}{2}(\widehat{\Delta}_t - \Delta_t)^2\{\widehat{\Delta}_t \geq 0\}] - \kappa\,p\,\mathbb{E}[\tfrac{1}{2}(\widehat{\Delta}_t - \Delta_t)^2\{\widehat{\Delta}_t < 0\}]$$

$$+ \kappa\,\mathbb{E}[\mathbb{E}[\beta_t\,|\widehat{\Delta}_t]\{\widehat{\Delta}_t \geq 0\}] + \kappa\,p\,\mathbb{E}[\mathbb{E}[\beta_t\,|\widehat{\Delta}_t]\{\widehat{\Delta}_t < 0\}] + \gamma p\,\mathbb{E}[\{\widehat{\Delta}_t < 0\}], \qquad (5)$$

where in the inequality we have dropped $p$ from factor $(1 - p)$ and combined the resulting terms $\mathbb{E}[R'_t\{\widehat{\Delta}_t < 0\}]$ and $\mathbb{E}[R'_t\{\widehat{\Delta}_t \geq 0\}]$ into $\mathbb{E}[R'_t]$. In turn, this term has been bounded as $\mathbb{E}[R'_t] \leq \frac{1}{|\widehat{\Delta}_t|}\,E[(\widehat{\Delta}_t - \Delta_t)^2] \leq \frac{1}{\gamma}\,E[(\widehat{\Delta}_t - \Delta_t)^2]$ by virtue of (2) with $\tau_t = 0$. At this point we work in the conditional space where $\widehat{\Delta}_t$ is given and distinguish the two cases $\widehat{\Delta}_t \geq 0$ and $\widehat{\Delta}_t < 0$. In the first case we have

$$(5) = \mathbb{E}[(\widehat{\Delta}_t - \Delta_t)^2\,|\widehat{\Delta}_t](1 - \kappa/2) + \kappa\mathbb{E}[\beta_t\,|\widehat{\Delta}_t] \leq 4(1 - \kappa/2) + \kappa\mathbb{E}[\beta_t\,|\widehat{\Delta}_t],$$

whereas in the second case we have

$$(5) = \mathbb{E}[(\widehat{\Delta}_t - \Delta_t)^2\,|\widehat{\Delta}_t](1 - \tfrac{1}{2}\kappa p) + \kappa p\mathbb{E}[\beta_t\,|\widehat{\Delta}_t] + \gamma p \leq 4(1 - \tfrac{1}{2}\kappa p) + \kappa p\mathbb{E}[\beta_t\,|\widehat{\Delta}_t] + \gamma p,$$

where in both cases we used $(\widehat{\Delta}_t - \Delta_t)^2 \leq 4$. We set $\kappa = 4\sqrt{n}$ and sum over $t = 1, \ldots, n$. Notice that $\sum_{t=1}^{n} D_t \leq \frac{a}{2}\|\boldsymbol{u}\| = \frac{a}{2}$ and that $\sum_{t=1}^{n} \beta_t \leq 2\ln\frac{|A_{n+1}|}{|A_1|} \leq 2\,N\ln\left(1 + \frac{n}{a\,N}\right)$ (e.g., [3], proof of Theorem 4.6 therein). After a few overapproximations (and taking the worst between the two cases $\widehat{\Delta}_t \geq 0$ and $\widehat{\Delta}_t < 0$) we obtain

$$\sum_{t=1}^{n} \gamma\mathbb{E}[R_t] \leq 2a\sqrt{n} + 8\sqrt{n}N\ln\left(1 + \frac{n}{a\,N}\right) + \frac{\gamma}{2}\sqrt{n} + \frac{\gamma}{2},$$

thereby concluding the proof. $\qquad\qquad\qquad\qquad\qquad\qquad\qquad\qquad\qquad\qquad\qquad\qquad\square$

## Footnotes

*The research was supported by the European Commission under the KerMIT Project No. IST-2001-25431.

[1]This parametrization requires the knowledge of $n$. It turns out one can remove this assumption at the cost of a slightly more involved proof.

## References

[1] Abe, N., and Long, P.M. (1999). Associative reinforcement learning using linear probabilistic concepts. In *Proc. ICML'99*, Morgan Kaufmann.

[2] Auer, P. (2000). Using Upper Confidence Bounds for Online Learning. In *Proc. FOCS'00*, IEEE, pages 270–279.

[3] Azoury, K., and Warmuth, M.K. (2001). Relative loss bounds for on-line density estimation with the exponential family of distributions, *Machine Learning*, 43:211–246.

[4] Censor, Y., and Lent, A. (1981). An iterative row-action method for interval convex programming. *Journal of Optimization Theory and Applications*, 34(3), 321–353.

[5] Cesa-Bianchi, N. (1999). Analysis of two gradient-based algorithms for on-line regression. *Journal of Computer and System Sciences*, 59(3):392–411.

[6] Cesa-Bianchi, N., Conconi, A., and Gentile, C. (2002). A second-order Perceptron algorithm. In *Proc. COLT'02*, pages 121–137. LNAI 2375, Springer.

[7] Cesa-Bianchi, N., Long, P.M., and Warmuth, M.K. (1996). Worst-case quadratic loss bounds for prediction using linear functions and gradient descent. *IEEE Trans. NN*, 7(3):604–619.

[8] Gavaldà, R., and Watanabe, O. (2001). Sequential sampling algorithms: Unified analysis and lower bounds. In *Proc. SAGA'01*, pages 173–187. LNCS 2264, Springer.

[9] Helmbold, D.P., Littlestone, N., and Long, P.M. (2000). Apple tasting. *Information and Computation*, 161(2):85–139.

[10] Herbster, M. and Warmuth, M.K. (1998). Tracking the best regressor, in *Proc. COLT'98*, ACM, pages 24–31.

[11] Hoeffding, W. (1963). Probability inequalities for sums of bounded random variables. *Journal of the American Statistical Association*, 58:13–30.

[12] Hoerl, A., and Kennard, R. (1970). Ridge regression: biased estimation for nonorthogonal problems. *Technometrics*, 12:55–67.

[13] Vapnik, V. (1998). *Statistical learning theory*. New York: J. Wiley & Sons.

[14] Voorhees, E., Harman, D. (2001). The tenth Text REtrieval Conference. TR 500-250, NIST.

[15] Vovk, V. (2001). Competitive on-line statistics. *International Statistical Review*, 69:213–248.
